# AN OPTIMIZATION NETWORK FOR MATRIX INVERSION

Ju-Seog Jang, Soo-Young Lee, and Sang-Yung Shin
Korea Advanced Institute of Science and Technology,
P.O. Box 150, Cheongryang, Seoul, Korea

## ABSTRACT

Inverse matrix calculation can be considered as an optimization. We have demonstrated that this problem can be rapidly solved by highly interconnected simple neuron-like analog processors. A network for matrix inversion based on the concept of Hopfield's neural network was designed, and implemented with electronic hardware. With slight modifications, the network is readily applicable to solving a linear simultaneous equation efficiently. Notable features of this circuit are potential speed due to parallel processing, and robustness against variations of device parameters.

## INTRODUCTION

Highly interconnected simple analog processors which mimic a biological neural network are known to excel at certain collective computational tasks. For example, Hopfield and Tank designed a network to solve the traveling salesman problem which is of the $np$-complete class,[1] and also designed an A/D converter of novel architecture[2] based on the Hopfield's neural network model.[3, 4] The network could provide good or optimum solutions during an elapsed time of only a few characteristic time constants of the circuit.

The essence of collective computation is the dissipative dynamics in which initial voltage configurations of neuron-like analog processors evolve simultaneously and rapidly to steady states that may be interpreted as optimal solutions. Hopfield has constructed the computational energy E (Liapunov function), and has shown that the energy function E of his network decreases in time when coupling coefficients are symmetric. At the steady state E becomes one of local minima.

In this paper we consider the matrix inversion as an optimization problem, and apply the concept of the Hopfield neural network model to this problem.

## CONSTRUCTION OF THE ENERGY FUNCTIONS

Consider a matrix equation $AV=I$, where $A$ is an input $n \times n$ matrix, $V$ is the unknown inverse matrix, and $I$ is the identity matrix. Following Hopfield we define $n$ energy functions $E_k$, $k=1, 2, ..., n$,

$$E_1 = (1/2)[(\sum_{j=1}^{n} A_{1j}V_{j1}-1)^2 + (\sum_{j=1}^{n} A_{2j}V_{j1})^2 + \cdots + (\sum_{j=1}^{n} A_{nj}V_{j1})^2]$$

$$E_2 = (1/2)[(\sum_{j=1}^{n} A_{1j}V_{j2})^2 + (\sum_{j=1}^{n} A_{2j}V_{j2}-1)^2 + \cdots + (\sum_{j=1}^{n} A_{nj}V_{j2})^2]$$

$$\cdots \cdots \cdots \cdots \cdots \cdots$$

$$E_n = (1/2)[(\sum_{j=1}^{n} A_{1j}V_{jn})^2 + (\sum_{j=1}^{n} A_{2j}V_{jn})^2 + \cdots + (\sum_{j=1}^{n} A_{nj}V_{jn} - 1)^2] \qquad (1)$$

where $A_{ij}$ and $V_{ij}$ are the elements of $i$th row and $j$th column of matrix $A$ and $V$, respectively. When $A$ is a nonsingular matrix, the minimum value (=zero) of each energy function is unique and is located at a point in the corresponding hyperspace whose coordinates are $\{ V_{1k}, V_{2k}, \cdots, V_{nk} \}$, $k = 1, 2, ..., n$. At this minimum value of each energy function the values of $V_{11}, V_{12}, ..., V_{nn}$ become the elements of the inverse matrix $A^{-1}$. When $A$ is a singular matrix the minimum value (in general, not zero) of each energy function is not unique and is located on a contour line of the minimum value. Thus, if we construct a model network in which initial voltage configurations of simple analog processors, called neurons, converge simultaneously and rapidly to the minimum energy point, we can say the network have found the *optimum* solution of matrix inversion problem. The optimum solution means that when $A$ is a nonsingular matrix the result is the inverse matrix that we want to know, and when $A$ is a singular matrix the result is a solution that is optimal in a least-square sense of Eq. (1).

## DESIGN OF THE NETWORK AND THE HOPFIELD MODEL

Designing the network for matrix inversion, we use the Hopfield model without inherent loss terms, that is,

$$\frac{du_{ik}}{dt} = -\frac{\partial}{\partial V_{ik}} E_k(V_{1k}, V_{2k}, \cdots, V_{nk})$$

$$V_{ik} = g_{ik}(u_{ik}), \qquad i, k = 1, 2, ..., n \qquad (2)$$

where $u_{ik}$ is the input voltage of $i$th neuron in the $k$th network, $V_{ik}$ is its output, and the function $g_{ik}$ is the input-output relationship. But the neurons of this scheme operate in all the regions of $g_{ik}$ differently from Hopfield's nonlinear 2-state neurons of associative memory models.[3,4]

From Eq. (1) and Eq. (2), we can define coupling coefficients $T_{ij}$ between $i$th and $j$th neurons and rewrite Eq. (2) as

$$\frac{du_{ik}}{dt} = -\sum_{j=1}^{n} T_{ij}V_{jk} + A_{ki} , \qquad T_{ij} = \sum_{l=1}^{n} A_{li}A_{lj} = T_{ji} ,$$

$$V_{ik} = g_{ik}(u_{ik}). \qquad (3)$$

It may be noted that $T_{ij}$ is independent of $k$ and only one set of hardware is needed for all $k$. The implemented network is shown in Fig. 1. The same set of hardware with bias levels, $\sum_{j=1}^{n} A_{ji}b_j$, can be used to solve a linear simultaneous

equation represented by **Ax=b** for a given vector **b**.

INPUT

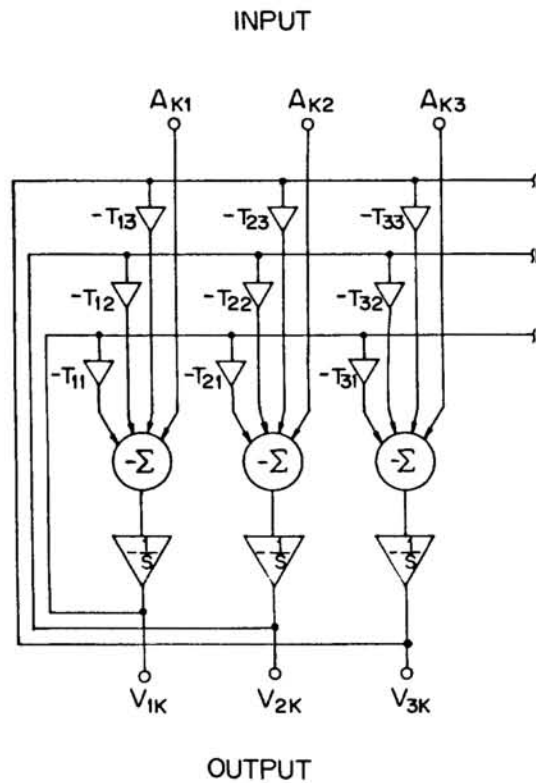

OUTPUT

Fig. 1. Implemented network for matrix inversion with externally
controllable coupling coefficients. Nonlinearity between
the input and the output of neurons is assumed to be
distributed in the adder and the integrator.

The application of the gradient Hopfield model to this problem gives the result
that is similar to the steepest descent method.[5] But the nonlinearity between the
input and the output of neurons is introduced. Its effect to the computational
capability will be considered next.

## CHARACTERISTICS OF THE NETWORK

For a simple case of 3×3 input matrices the network is implemented with
electronic hardware and its dynamic behavior is simulated by integration of the
Eq. (3). For nonsingular input matrices, exact realization of $T_{ij}$ connection and
bias $A_{ki}$ is an important factor for calculation accuracy, but the initial condition
and other device parameters such as steepness, shape and uniformity of $g_{ik}$ are
not. Even a complex $g_{ik}$ function shown in Fig. 2 can not affect the computa-
tional capability. Convergence time of the output state is determined by the
characteristic time constant of the circuit. An example of experimental results is
shown in Fig. 3. For singular input matrices, the converged output voltage confi-
guration of the network is dependent upon the initial state and the shape of $g_{ik}$.

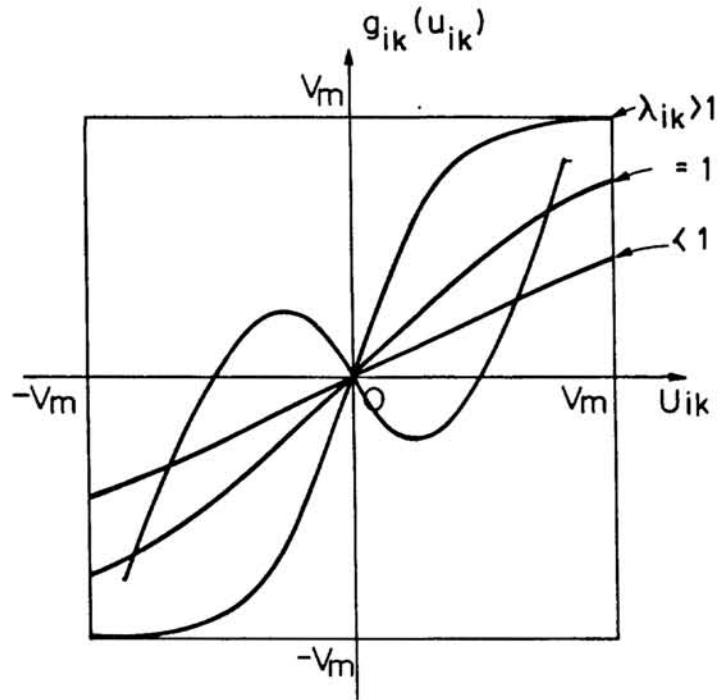

Fig. 2. $g_{ik}$ functions used in computer simulations where $\lambda_{ik}$ is the steepness of sigmoid function $tanh(\lambda_{ik}u_{ik})$.

input matrix $A = \begin{pmatrix} 1 & 2 & 1 \\ -1 & 1 & 1 \\ 1 & 0 & -1 \end{pmatrix}$ (cf) $A^{-1} = \begin{pmatrix} 0.5 & -1 & -0.5 \\ 0 & 1 & 1 \\ 0.5 & -1 & -1.5 \end{pmatrix}$

output matrix $V = \begin{pmatrix} 0.50 & -0.98 & -0.49 \\ 0.02 & 0.99 & 1.00 \\ 0.53 & -0.98 & -1.50 \end{pmatrix}$

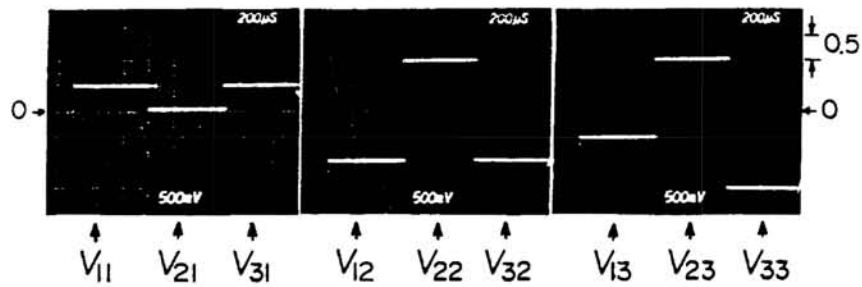

$V_{11}$ $V_{21}$ $V_{31}$    $V_{12}$ $V_{22}$ $V_{32}$    $V_{13}$ $V_{23}$ $V_{33}$

Fig. 3. An example of experimental results

## COMPLEXITY ANALYSIS

By counting operations we compare the neural net approach with other well-known methods such as Triangular-decomposition and Gauss-Jordan elimination.[6]

(1) Triangular-decomposition or Gauss-Jordan elimination method takes $O(8n^3/3)$ multiplications/divisions and additions for large $n \times n$ matrix inversion, and $O(2n^3/3)$ multiplications/divisions and additions for solving the linear simultaneous equation $\mathbf{Ax=b}$.

(2) The neural net approach takes the number of operations required to calculate $T_{ij}$ (nothing but matrix-matrix multiplication), that is, $O(n^3/2)$ multiplications and additions for both matrix inversion and solving the linear simultaneous equation. And the time required for output stablization is about a few times the characteristic time constant of the network. The calculation of coupling coefficients can be directly executed without multiple iterations by a specially designed optical matrix-matrix multiplier,[7] while the calculation of bias values in solving a linear simultaneous equation can be done by an optical vector-matrix multiplier.[8] Thus, this approach has a definite advantage in potential calculation speed due to global interconnection of simple parallel analog processors, though its calculation accuracy may be limited by the nature of analog computation. A large number of controllable $T_{ij}$ interconnections may be easily realized with optoelectronic devices.[9]

## CONCLUSIONS

We have designed and implemented a matrix inversion network based on the concept of the Hopfield's neural network model. This network is composed of highly interconnected simple neuron-like analog processors which process the information in parallel. The effect of sigmoid or complex nonlinearities on the computational capability is unimportant in this problem. Steep sigmoid functions reduce only the convergence time of the network. When a nonsingular matrix is given as an input, the network converges spontaneously and rapidly to the correct inverse matrix regardless of initial conditions. When a singular matrix is given as an input, the network gives a stable optimum solution that depends upon initial conditions of the network.

## REFERENCES

1. J. J. Hopfield and D. W. Tank, Biol. Cybern. 52, 141 (1985).
2. D. W. Tank and J. J. Hopfield, IEEE Trans. Circ. Sys. CAS-33, 533 (1986).
3. J. J. Hopfield, Proc. Natl. Acad. Sci. U.S.A. 79, 2554 (1982).
4. J. J. Hopfield, Proc. Natl. Acad. Sci. U.S.A. 81, 3088 (1984).
5. G. A. Bekey and W. J. Karplus, Hybrid Computation (Wiley, 1968), P. 244.
6. M. J. Maron, Numerical Analysis: A Practical Approach (Macmillan, 1982), p. 138.
7. H. Nakano and K. Hotate, Appl. Opt. 26, 917 (1987).
8. J. W. Goodman, A. R. Dias, and I. M. Woody, Opt. Lett. 2, 1 (1978).
9. J. W. Goodman, F. J. Leonberg, S-Y. Kung, and R. A. Athale, IEEE Proc. 72, 850 (1984).
